# Spectral Regularization for Support Estimation

**Ernesto De Vito**
DSA, Univ. di Genova, and
INFN, Sezione di Genova, Italy
devito@dima.ungie.it

**Lorenzo Rosasco**
CBCL - MIT, - USA, and
IIT, Italy
lrosasco@mit.edu

**Alessandro Toigo**
Politec. di Milano, Dept. of Math., and
INFN, Sezione di Milano, Italy
toigo@ge.infn.it

## Abstract

In this paper we consider the problem of learning from data the support of a probability distribution when the distribution *does not* have a density (with respect to some reference measure). We propose a new class of regularized spectral estimators based on a new notion of reproducing kernel Hilbert space, which we call *"completely regular"*. Completely regular kernels allow to capture the relevant geometric and topological properties of an arbitrary probability space. In particular, they are the key ingredient to prove the universal consistency of the spectral estimators and in this respect they are the analogue of universal kernels for supervised problems. Numerical experiments show that spectral estimators compare favorably to state of the art machine learning algorithms for density support estimation.

## 1 Introduction

In this paper we consider the problem of estimating the support of an arbitrary probability distribution and we are more broadly motivated by the problem of learning from complex high dimensional data. The general intuition that allows to tackle these problems is that, though the initial representation of the data is often very high dimensional, in most situations the data are not uniformly distributed, but are in fact confined to a small (possibly low dimensional) region. Making such an intuition rigorous is the key towards designing effective algorithms for high dimensional learning.
The problem of estimating the support of a probability distribution is of interest in a variety of applications such as anomaly/novelty detection [8], or surface modeling [16]. From a theoretical point of view the problem has been usually considered in the setting where the probability distribution has a density with respect to a known measure (for example the Lebesgue measure in $\mathbb{R}^d$ or the volume measure on a manifold). Among others we mention [22, 5] and references therein. Algorithms inspired by Support Vector Machine (SVM), often called one-class SVM are have been proposed see [17, 20] and references therein. Another kernel method, related to the one we discuss in this paper, is presented in [11]. More generally one of the main approaches to learning from high dimensional is the one considered in manifold learning. In this context the data are assumed to lie on a low dimensional Riemannian sub-manifold embedded (that is represented) in a high dimensional Euclidean space. This framework inspired algorithms to solve a variety of problems such as: semisupervised learning [3], clustering [23], data parameterization/dimensionality reduction [15, 21], to name a few. The basic assumption underlying manifold learning is often too restrictive to describe real data and this motivates considering other models, such as the setting where the data are assumed to be *essentially* concentrated around a low dimensional manifold as in [12], or can be modeled as samples from a metric space as in [10].

In this paper we consider a general scenario (see [18]) where the underlying model is a probability space $(X, \rho)$ and we are given a (*similarity*) function $K$ which is a reproducing kernel. The available training set is an i.i.d sample $x_1, \ldots, x_n \sim \rho$. The geometry (and topology) in $(X, \rho)$ is defined by the kernel $K$. While this framework is abstract and poses new challenges, by assuming the similarity function to be a reproducing kernel we can make full use of the good computational properties of kernel methods and the powerful theory of reproducing kernel Hilbert spaces (RKHS) [2]. Interestingly, the idea of using a reproducing kernel $K$ to construct a metric on a set $X$ is originally due to Schoenberg (see for example [4]).

Broadly speaking, in this setting we consider the problem of *finding a model of the smallest region $X_\rho$ containing all the data*. A rigorous formalization of this problem requires: 1) defining the region $X_\rho$, 2) specifying the sense in which we model $X_\rho$. This can be easily done if the probability distribution has density $p$ with respect to a known measure, in fact $X_\rho = \{x \in X \ : \ p(x) > 0\}$, but is otherwise a challenging question for a general distribution. Intuitively, $X_\rho$ can be thought of as the region where the distribution is concentrated, that is $\rho(X_\rho) = 1$. However, there are many different sets having this property. If $X$ is $\mathbb{R}^d$ (in fact any topological space), a natural candidate to define the region of interest, is the notion of *support* of a probability distribution– defined as the intersection of the closed subsets $C$ of $X$, such that $\rho(C) = 1$. In an arbitrary probability space the support of the measure is not well defined since no topology is given.

The reproducing kernel $K$ provides a way to solve this problem and also suggests a possible approach to model $X_\rho$. The first idea is to use the fact that under mild assumptions the kernel defines a metric on $X$ [18], so that the concept of closed set, hence that of support, is well defined. The second idea is to use the kernel to construct a function $F_\rho$ such that the level set corresponding to one is exactly the support $X_\rho$– in this case we say that the RKHS associated to $K$ *separates* the support $X_\rho$. By doing this we are in fact imposing an assumption on $X_\rho$: given a kernel $K$, we can only separate certain sets. More precisely, our contribution is two-fold.

- We prove that $F_\rho$ is uniquely defined by the null space of the integral operator associated to $K$. Given that the integral operator (and its spectral properties) can be approximated studying the kernel matrix on a sample, this result suggests a way to estimate the support empirically. However, a further complication arises from the fact that in general zero is not an isolated point of the spectrum, so that the estimation of a null space is an ill-posed problem (see for example [9]). Then, a regularization approach is needed in order to find a stable (hence generalizing) estimator. In this paper, we consider a spectral estimator based on a spectral regularization strategy, replacing the kernel matrix with its regularized version (Tikhonov regularization being one example).

- We introduce the notion of *completely regular RKHS*, that answer positively to the question whether there exist kernels that can separate the support of *any* distribution. Examples of completely regular kernels are presented and results suggesting how they can be constructed are given. The concept of completely regular RKHS plays a role similar to the concept of universal kernels in supervised learning, for example see [19].

Finally, given the above results, we show that the regularized spectral estimator enjoys a universal consistency property: the correct support can be asymptotically recovered for *any* problem (that is any probability distribution).

The plan of the paper is as follows. In Section 2 we introduce the notion of completely regular kernels and their basic properties. In Section 3 we present the proposed regularized algorithms. In Section 4 and 5 we provide a theoretical and empirical analysis, respectively. Proofs and further development can be found in the supplementary material.

## 2 Completely regular reproducing kernel Hilbert spaces

In this section we introduce the notion of a completely regular reproducing kernel Hilbert space. Such a space defines a geometry on a measurable space $X$ which is compatible with the measurable structure. Furthermore it shows how to define a function $F$ such that the one level set is the support of the probability distribution. The function is determined by the spectral projection associated with the null eigenvalue of the integral operator defined by the reproducing kernel. All the proofs of this section are reported in the supplementary material.

We assume $X$ to be a measurable space with a probability measure $\rho$. We fix a complex[1] reproducing kernel Hilbert space $\mathcal{H}$ on $X$ with a reproducing kernel $K : X \times X \rightarrow \mathbb{C}$ [2]. The scalar product and the norm are denoted by $\langle \cdot, \cdot \rangle$, linear in the first argument, and $\|\cdot\|$, respectively. For all $x \in X$, $K_x \in \mathcal{H}$ denotes the function $K(\cdot, x)$. For each function $f \in \mathcal{H}$, the reproducing property $f(x) = \langle f, K_x \rangle$ holds for all $x \in X$. When different reproducing kernel Hilbert spaces are considered, we denote by $\mathcal{H}_K$ the reproducing kernel Hilbert space with reproducing kernel $K$. Before giving the definition of completely regular RKHS, which is the key concept presented in this section, we need some preliminary definitions and results.

**Definition 1.** *A subset $C \subset X$ is* separated *by $\mathcal{H}$, if, for any $x_0 \notin C$, there exists $f \in \mathcal{H}$ such that*

$$f(x_0) \neq 0 \quad and \quad f(x) = 0 \qquad \forall x \in C. \tag{1}$$

For example, if $X = \mathbb{R}^d$ and $\mathcal{H}$ is the reproducing kernel Hilbert space with linear kernel $K(x, t) = x \cdot t$, the sets separated by $\mathcal{H}$ are precisely the hyperplanes containing the origin. In Eq. (1) the function $f$ depends on $x_0$ and $C$, but Proposition 1 below will show that there is a function, possibly not in $\mathcal{H}$, whose one level set is precisely $C$ ( if $K(x, x) = 1$ ). Note that in [19] a different notion of *separating property* is given.

We need some further notation. For any set $C$, let $P_C : \mathcal{H} \rightarrow \mathcal{H}$ be the orthogonal projection onto the closure of the linear space generated by $\{K_x \mid x \in C\}$, so that $P_C^2 = P_C$, $P_C^* = P_C$ and

$$\ker P_C = \{K_x \mid x \in C\}^\perp = \{f \in \mathcal{H} \mid f(x) = 0, \ \forall x \in C\}.$$

Moreover let $F_C : X \rightarrow \mathbb{C}$ be defined by $F_C(x) = \langle P_C K_x, K_x \rangle$.

**Proposition 1.** *For any subset $C \subset X$, the following facts are equivalent*

  *(i) the set $C$ is separated by $\mathcal{H}$;*

  *(ii) for all $x \notin C$, $K_x \notin \operatorname{Ran} P_C$;*

  *(iii) $C = \{x \in X \mid F_C(x) = K(x, x)\}$.*

*If one of the above conditions is satisfied, then $K(x, x) \neq 0 \qquad \forall x \notin C$.*

A natural and minimal requirement on $\mathcal{H}$ is to be able to separates any pairs of distinct points and this implies that $K_x \neq K_t$ if $x \neq t$ and $K(x, x) \neq 0$. The first condition ensures the metric given by

$$d_K(x, y) = \|K_x - K_t\| \qquad x, t \in X. \tag{2}$$

to be well defined. Then $(X, d_K)$ is a metric space and the sets separated by $\mathcal{H}$ are always $d_K$-closed, see Prop. 2 below. This last property is not enough to ensure that we can evaluate $\rho$ on the set separated by RKHS $\mathcal{H}$. In fact the $\sigma$-algebra generated by the metric $d$ might not be contained in the $\sigma$-algebra on $X$. The next result shows that assuming the kernel to be measurable is enough to solve this problem.

**Proposition 2.** *Assume that $K_x \neq K_t$ if $x \neq t$, then the sets separated by $\mathcal{H}$ are closed with respect to $d_K$. Moreover, if $\mathcal{H}$ is separable and the kernel is measurable, then the sets separated by $\mathcal{H}$ are measurable.*

Given the above premises, the following is the key definition that characterizes the reproducing kernel Hilbert spaces which are able to separate the *largest* family of subsets of $X$.

**Definition 2** (Completely Regular RKHS). *A reproducing kernel Hilbert space $\mathcal{H}$ with reproducing kernel $K$ such that $K_x \neq K_t$ if $x \neq t$ is called completely regular if $\mathcal{H}$ separates all the subsets $C \subset X$ which are closed with respect to the metric (2).*

The term *completely regular* is borrowed from topology, where a topological space is called completely regular if, for any closed subset $C$ and any point $x_0 \notin C$, there exists a continuous function $f$ such that $f(x_0) \neq 0$ and $f(x) = 0$ for all $x \in C$. In the supplementary material, several examples of completely regular reproducing kernel Hilbert spaces are given, as well as a discussion on how such spaces can be constructed. A particular case is when $X$ is already a metric space with a distance

function $d_X$. If $K$ is continuous with respect to $d_X$, the assumption of complete regularity forces the metrics $d_K$ and $d_X$ to have the same closed subsets. Then, the supports defined by $d_K$ and $d_X$ are the same. Furthermore, since the closed sets of $X$ are independent of $\mathcal{H}$, the complete regularity of $\mathcal{H}$ can be proved by showing that a suitable family of *bump*[2] functions is contained in $\mathcal{H}$.

**Corollary 1.** *Let $X$ be a separable metric space with respect to a metric $d_X$. Assume that the kernel $K$ is a continuous function with respect to $d_X$ and that the space $\mathcal{H}$ separates every subset $C$ which is closed with respect to $d_X$. Then*

   *(i) The space $\mathcal{H}$ is separable and $K$ is measurable with respect to the Borel $\sigma$-algebra generated by $d_X$.*

   *(ii) The metric $d_K$ defined by (2) is equivalent to $d_X$, that is, a set is closed with respect to $d_K$ if and only if it is closed with respect to $d_X$.*

   *(iii) The space $\mathcal{H}$ is completely regular.*

As a consequence of the above result, many classical reproducing kernel Hilbert spaces are completely regular. For example, if $X = \mathbb{R}^d$ and $\mathcal{H}$ is the Sobolev space of order $s$ with $s > d/2$, then $\mathcal{H}$ is completely regular. This is due to the fact that the space of smooth compactly supported functions is contained in $\mathcal{H}$. In fact, a standard result of analysis ensures that, for any closed set $C$ and any $x_0 \notin C$ there exists a smooth bump function such that $f(x_0) = 1$ and its support is contained in the complement of $C$. Interestingly enough, if $\mathcal{H}$ is the reproducing kernel Hilbert space with the Gaussian kernel, it is known that the elements of $\mathcal{H}$ are analytic functions, see Cor. 4.44 in [19]. Clearly $\mathcal{H}$ can not be completely regular. Indeed, if $C$ is a closed subset of $\mathbb{R}^d$ with not empty interior and $f \in \mathcal{H}$ is such that $f(x) = 0$ for all $x \in C$, a standard result of complex analysis implies that $f(x) = 0$ for every $x \in \mathbb{R}^d$. Finally, the next result shows that the reproducing kernel can be normalized to one on the diagonal under the mild assumption that $K(x, x) \neq 0$ for all $x \in X$.

**Lemma 1.** *Assume that $K(x, x) > 0$ for all $x \in X$. Then the reproducing kernel Hilbert space with the normalized kernel $K'(x, t) = \dfrac{K(x, t)}{\sqrt{K(x, x) K(t, t)}}$ separates the same sets as $\mathcal{H}$.*

Finally we briefly mention some examples and refer to the supplementary material for further developments. In particular, we prove that both the Laplacian kernel $K(x, y) = e^{-\frac{\|x-y\|_2}{\sqrt{2}\sigma}}$ and $\ell_1$-exponential kernel $K(x, y) = e^{-\frac{\|x-y\|_1}{\sqrt{2}\sigma}}$ defined on $\mathbb{R}^d$ are completely regular for any $\sigma > 0$ and $d \in \mathbb{N}$.

## 3    Spectral Algorithms for Learning the Support

In this section, we first discuss our framework and our main assumptions. Then we present the proposed regularized spectral algorithms.

Motivated by the results in the previous section, we describe our framework which is given by a triple $(X, \rho, K)$. We consider a probability space $(X, \rho)$ and a training set $\mathbf{x} = (x_1 \ldots, x_n)$ sampled i.i.d. with respect to $\rho$. Moreover we consider a reproducing kernel $K$ satisfying the following assumption.

**Assumption 1.** *The reproducing kernel $K$ is measurable and $K(x, x) = 1$, for all $x \in X$. Moreover $K$ defines a completely regular and separable RKHS $\mathcal{H}$.*

We endow $X$ with the metric $d_K$ defined in (2), so that $X$ becomes a separable metric space. The assumption of complete regularity ensures that any closed subset is separated by $\mathcal{H}$ and, hence, is measurable by Prop. 2. Then we can define the support $X_\rho$ of the measure $\rho$, as the intersection of all the closed sets $C \subset X$, such that $\rho(C) = 1$. Clearly $X_\rho$ is closed and $\rho(X_\rho) = 1$ (note that this last property depends on the separability of $X$, hence of $\mathcal{H}$).

Summarizing the key result in the previous section, under the above assumptions, $X_\rho$ is the one level set of the function $F_\rho : X \to [0, 1]$

$$F_\rho(x) = \langle P_\rho K_x, K_x \rangle ,$$

where $P_\rho$ is a short notation for $P_{X_\rho}$. Since $F_\rho$ depends on the unknown measure $\rho$, in practice it cannot be explicitly calculated. To design an effective empirical estimator we develop a novel characterization of the support of an arbitrary distribution that we describe in the next section.

## 3.1 A New Characterization of the Support

The key observation towards defining a learning algorithm to estimate $X_\rho$ it is that the projection $P_\rho$ can be expressed in terms of the integral operator defined by the kernel $K$.
To see this, for all $x \in X$, let $K_x \otimes K_x$ denote the rank one positive operator on $\mathcal{H}$, given by

$$(K_x \otimes K_x)(f) = \langle f, K_x \rangle K_x = f(x)K_x \qquad f \in \mathcal{H}.$$

Moreover, let $T : \mathcal{H} \to \mathcal{H}$ be the linear operator defined as

$$T = \int_X K_x \otimes K_x d\rho(x),$$

where the integral converges in the Hilbert space of Hilbert-Schmidt operators on $\mathcal{H}$ (see for example [7] for the proof). Using the reproducing property in $\mathcal{H}$ [2], it is straightforward to see that $T$ is simply the integral operator with kernel $K$ with domain and range in $\mathcal{H}$.
Then, one can easily see that the null space of $T$ is precisely $(I - P_\rho)\mathcal{H}$, so that

$$P_\rho = T^\dagger T, \tag{3}$$

where $T^\dagger$ is the pseudo-inverse of $T$ (see for example [9]). Hence

$$F_\rho(x) = \langle T^\dagger T K_x, K_x \rangle.$$

Observe that in general $K_x$ does not belong to the domain of $T^\dagger$ and, if $\theta$ denotes the Heaviside function with $\theta(0) = 0$, then spectral theory gives that $P_\rho = T^\dagger T = \theta(T)$. The above observation is crucial as it gives a new characterization of the support of $\rho$ in terms of the null space of $T$ and the latter can be estimated from data.

## 3.2 Spectral Regularization Algorithms

Finally, in this section, we describe how to construct an estimator $F_n$ of $F_\rho$. As we mentioned above, Eq. (3) suggests a possible way to learn the projection from finite data. In fact, we can consider the empirical version of the integral operator associated to $K$ which is simply defined by

$$T_n = \frac{1}{n} \sum_{i=1}^{n} K_{x_i} \otimes K_{x_i}.$$

The latter operator is an unbiased estimator of $T$. Indeed, since $K_x \otimes K_x$ is a bounded random variable into the separable Hilbert space of Hilbert-Schmidt operators, one can use concentration inequalities for random variables in Hilbert spaces to prove that

$$\lim_{n \to +\infty} \frac{\sqrt{n}}{\log n} \|T - T_n\|_{\text{HS}} = 0 \qquad \text{almost surely,} \tag{4}$$

where $\|\cdot\|_{\text{HS}}$ is the Hilbert-Schmidt norm (see for example [14] for a short proof). However, in general $T_n^\dagger T_n$ does non converge to $T^\dagger T$ since 0 is an accumulation point of the spectrum of $T$ or, equivalently, since $T^\dagger$ is not a bounded operator. Hence, a regularization approach is needed.
In this paper we study a spectral filtering approach which replaces $T_n^\dagger$ with an approximation $g_\lambda(T_n)$ obtained *filtering out* the components corresponding to the small eigenvalues of $T_n$. The function $g_\lambda$ is defined by spectral calculus. More precisely if $T_n = \sum_j \sigma_j v_j \otimes v_j$ is a spectral decomposition of $T_n$, then $g_\lambda(T_n) = \sum_j g_\lambda(\sigma_j)v_j \otimes v_j$. Spectral regularization defined by linear filters is classical in the theory of inverse problems [9]. Intuitively, $g_\lambda(T_n)$ is an approximation of the generalized inverse $T_n^\dagger$ and it is such that the approximation gets better, but the condition number of $g_\lambda(T_n)$ gets worse as $\lambda$ decreases. More formally these properties are captured by the following set of conditions.

**Assumption 2.** *For $\sigma \in [0, 1]$, let $r_\lambda(\sigma) := \sigma g_\lambda(\sigma)$, then*

- $r_\lambda(\sigma) \in [0, 1], \quad \forall \lambda > 0,$

- $\lim_{\lambda \to 0} r_\lambda(\sigma) = 1, , \quad \forall \sigma > 0$

- $|r_\lambda(\sigma) - r_\lambda(\sigma')| \le L_\lambda |\sigma - \sigma'|, \forall \lambda > 0$, *where $L_\lambda$ is a positive constant depending on $\lambda$.*

Examples of algorithms that fall into the above class include iterative methods– akin to boosting $g_\lambda(\sigma) = \sum_{k=0}^{m_\lambda}(1-\sigma)^k$, spectral cut-off $g_\lambda(\sigma) = \frac{1}{\sigma}\mathbf{1}_{\sigma > \lambda}(\sigma) + \frac{1}{\lambda}\mathbf{1}_{\sigma \le \lambda}(\sigma)$, and Tikhonov regularization $g_\lambda(\sigma) = \frac{1}{\sigma + \lambda}$. We refer the reader to [9] for more details and examples, and, given the space constraints, will focus mostly on Tikhonov regularization in the following.

For a chosen filter, the regularized empirical estimator of $F_\rho$ can be defined by

$$F_n(x) = \langle g_\lambda(T_n)T_n K_x, K_x \rangle. \tag{5}$$

One can see that that the computation of $F_n$ reduces to solving a simple finite dimensional problem involving the empirical kernel matrix defined by the training data. Towards this end, it is useful to introduce the sampling operator $S_n : \mathcal{H} \to \mathbb{C}^n$ defined by $S_n f = (f(x_1), \ldots, f(x_n))$, $f \in \mathcal{H}$, which can be interpreted as the restriction operator which evaluates functions in $\mathcal{H}$ on the training set points. The adjoint $S_n^* : \mathbb{C}^n \to \mathcal{H}$ of $S_n$ is given by $S_n^*\alpha = \sum_{i=1}^n \alpha_i K_{x_i}$, $\alpha = (\alpha_1, \ldots, \alpha_n) \in \mathbb{C}^n$, and can be interpreted as the out-of-sample extension operator. A simple computation shows that $T_n = \frac{1}{n}S_n^*S_n$ and $S_n S_n^* = \mathbf{K}_n$ is the $n$ by $n$ kernel matrix, where the $(i, j)$-entry is $K(x_i, x_j)$. Then it is easy to see that $g_\lambda(T_n)T_n = g_\lambda(S_n^*S_n/n)S_n^*S_n/n = \frac{1}{n}S_n^* g_\lambda(\mathbf{K}_n/n)S_n$, so that

$$F_n(x) = \frac{1}{n}\mathbf{k}_x{}^T g_\lambda(\mathbf{K}_n/n)\mathbf{k}_x, \tag{6}$$

where $\mathbf{k}_x$ is the $n$-dimensional column vector $\mathbf{k}_x = S_n K_x = (K(x_1, x), \ldots, K(x_n, x))$. Note that Equation (6) plays the role of a representer theorem for the spectral estimator, in the sense that it reduces the problem of finding an estimator in an infinite dimensional space to a finite dimensional problem.

## 4 Theoretical Analysis: Universal Consistency

In this section we study the consistency property of spectral estimators. All the proofs of this section are reported in the supplementary material. We prove the results only for the filter corresponding to the classical Tikhonov regularization though the same results hold for the class of spectral filters described by Assumption 2. To study the consistency of the methods we need to choose an appropriate performance measure to compare $F_n$ and $F_\rho$. Note that there is no natural notion of *risk*, since we have to compute the function *on* and *off* the support. Also note that standard metric used for support estimation (see for example [22, 5]) cannot be used in our analsys since they rely on the existence of a reference measure $\mu$ (usually the Lebesgue measure) and the assumption that $\rho$ is absolutely continuous with respect to $\mu$.

The following preliminary result shows that we can control the convergence of the Tikhonov estimator $F_n$, defined by $g_\lambda(T) = (T_n + \lambda_n I)^{-1}$, to $F_\rho$ uniformly on any compact set of $X$, provided a suitable sequence $\lambda_n$.

**Theorem 1.** *Let $F_n$ be the estimator defined by Tikhonov regularization and choose a sequence $\lambda_n$ so that*

$$\lim_{n \to \infty} \lambda_n = 0 \quad and \quad \limsup_{n \to \infty} \frac{\log n}{\lambda_n \sqrt{n}} < +\infty, \tag{7}$$

*then*

$$\lim_{n \to +\infty} \sup_{x \in C} |F_n(x) - F_\rho(x)| = 0, \qquad almost\ surely, \tag{8}$$

*for every compact subset $C$ of $X$*

We add three comments. First, we note that, as we mentioned before, Tikhonov regularization can be replaced by a large class of filters. Second, we observe that a natural choice would be the regularization defined by kernel PCA [11], which corresponds to truncating the generalized inverse of the kernel matrix at some cutoff parameter $\lambda$. However, one can show that, in general, in this case it is not possible to choose $\lambda$ so that the sample error goes to zero. In fact, for KPCA the sample error depends on the gap between the $M$-th and the $M + 1$-th eigenvalue of $T$ [1], where $M$-th and $M + 1$-th are the eigenvalues around the cutoff parameter. Such a gap can go to zero with an

arbitrary rate so that there exists *no* choice of the cut-off parameter ensuring convergence to zero of the sample error. Third, we note that the uniform convergence of $F_n$ to $F_\rho$ on compact subsets *does not* imply the convergence of the level sets of $F_n$ to the corresponding level sets of $F_\rho$, for example with respect to the standard Hausdorff distance among closed subsets. In practice to have an effective decision rule, an off-set parameter $\tau_n$ can be introduced and the level set is replaced by $X_n = \{x \in X \mid F_n(x) \geq 1 - \tau_n\}$ – recall that $F_n$ takes values in $[0, 1]$. The following result will show that for a suitable choice of $\tau_n$ the Hausdorff distance between $X_n \cap C$ and $X_\rho \cap C$ goes to zero for all compact sets $C$. We recall that the Hausdorff distance between two subsets $A, B \subset X$ is

$$\mathrm{d}_H(A, B) = \max\{\sup_{a \in A} d_K(a, B), \sup_{b \in B} d_K(b, A)\}$$

**Theorem 2.** *If the sequence $(\tau_n)_{n \in \mathbb{N}}$ converges to zero in such a way that*

$$\limsup_{n \to \infty} \frac{\sup_{x \in C} |F_n(x) - F_\rho(x)|}{\tau_n} \leq 1, \qquad almost\ surely \qquad (9)$$

*then,*

$$\lim_{n \to +\infty} \mathrm{d}_H(X_n \cap C, X_\rho \cap C) = 0 \qquad almost\ surely,$$

*for any compact subset $C$.*

We add two comments. First, it is possible to show that, if the (normalized) kernel $K$ is such that $\lim_{x' \to \infty} K_x(x') = 0$ for any $x \in X$ – as it happens for the Laplacian kernel, then Theorems 1 and 2 also hold by choosing $C = X$. Second, note that the choice of $\tau_n$ depends on the rate of convergence of $F_n$ to $F_\rho$ which will itself depend on some a-priori assumption on $\rho$. Developing learning rates and finite sample bound is a key question that we will tackle in future work.

## 5    Empirical Analysis

In this section we describe some preliminary experiments aimed at testing the properties and the performances of the proposed methods both on simlauted and real data. Again for space constraints we will only discuss spectral algorithms induced by Tikhonov regularization. Note that while computations can be made efficient in several ways, we consider a simple algorithmic protocol and leave a more refined computational study for future work. Following the discussion in the last section, Tikhonov regularization defines an estimator $F_n(x) = \mathbf{k}_x{}^T(\mathbf{K}_n + n\lambda I)^{-1}\mathbf{k}_x$ and a point is labeled as belonging to the support if $F_n(x) \geq 1 - \tau$. The computational cost for the algorithm is, in the worst case, of order $n^3$, like standard regularized least squares, for training and order $Nn^2$ if we have to predict the value of $F_n$ at $N$ test points. In practice, one has to choose a good value for the regularization parameter $\lambda$ and this requires computing multiple solutions, a so called *regularization path*. As noted in [13], if we form the inverse using the eigendecomposition of the kernel matrix the price of computing the full regularization path is essentially the same as that of computing a single solution (note that the cost of the eigen-decomposition of $\mathbf{K}_n$ is also of order $n^3$ though the constant is worse). This is the strategy that we consider in the following. In our experiments we considered two data-sets the MNIST data-set and the CBCL face database. For the digits we considered a reduced set consisting of a training set of 5000 images and a test set of 1000 images. In the first experiment we trained on 500 images for the digit 3 and tested on 200 images of digits 3 and 8. Each experiment consists of training on one class and testing on two different classes and was repeated for 20 trials over different training set choices. The performance is evaluated computing ROC curve (and the corresponding AUC value) for varying $\tau, \tau', \tau''$. For all our experiments we considered the Laplacian kernel. Note that, in this case the algorithm requires to choose 3 parameters: the regularization parameter $\lambda$, the kernel width $\sigma$ and the threshold $\tau$. In supervised learning cross validation is typically used for parameter tuning, but cannot be used in our setting since support estimation is an unsupervised problem. Then, we considered the following heuristics. The kernel width is chosen as the median of the distribution of distances of the $K$-th nearest neighbor of each training set point for $K = 10$. Fixed the kernel width, we choose regularization parameter in correspondence of the maximum curvature in the eigenvalue behavior– see Figure 1, the rational being that after this value the eigenvalues are relatively small. For comparison we considered a Parzen window density estimator and one-class SVM (1CSVM )as implemented by [6]. For the Parzen window estimator we used the same kernel used in the spectral algorithm, that is the Laplacian kernel and use the

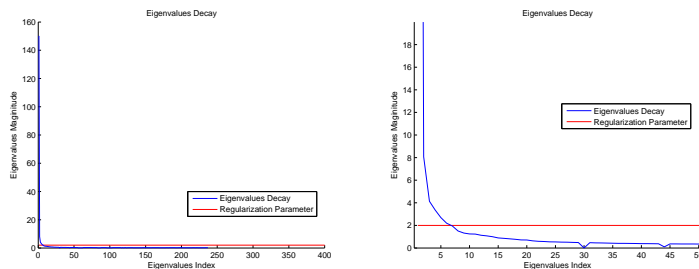

Figure 1: Decay of the eigenvalues of the kernel matrix ordered in decreasing magnitude and corresponding regularization parameter (Left) and a detail of the first $50$ eigenvalues (Right).

same width used in our estimator. Given a kernel width an estimate of the probability distribution is computed and can be used to estimate the support by fixing a threshold $\tau'$. For the one-class SVM we considered the Gaussian kernel, so that we have to fix the kernel width and a regularization parameter $\nu$. We fix the kernel width to be the same used by our estimator and fixed $\nu = 0.9$. For the sake of comparison, also for one-class SVM we considered a varying offset $\tau''$. The ROC curves on the different tasks are reported (for one of the trial) in Figure 2, Left. The mean and standard deviation of the AUC for the 3 methods is reported in Table 5. Similar experiments were repeated considering other pairs of digits, see Table 5. Also in the case of the CBCL data sets we considered a reduced data-set consisting of $472$ images for training and other $472$ for test. On the different test performed on the Mnist data the spectral algorithm always achieves results which are better- and often substantially better - than those of the other methods. On the CBCL dataset SVM provides the best result, but spectral algorithm still provides a competitive performance.

## 6 Conclusions

In this paper we presented a new approach to estimate the support of an arbitrary probability distribution. Unlike previous work we drop the assumption that the distribution has a density with respect to a (known) reference measure and consider a general probability space. To overcome this problem we introduce a new notion of RKHS, that we call completely regular, that captures the relevant geometric properties of the probability distribution. Then, the support of the distribution can be characterized as the null space of the integral operator defined by the kernel and can be estimated using a spectral filtering approach. The proposed estimators are proven to be universally consistent and have good empirical performances on some benchmark data-sets. Future work will be devoted

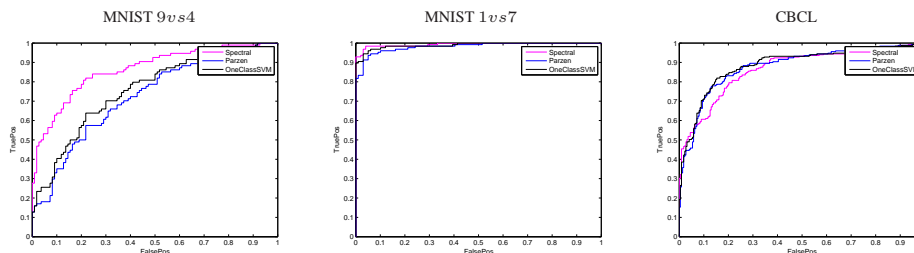

Figure 2: ROC curves for the different estimator in three different tasks: digit $9$vs $4$ Left, digit $1$vs $7$ Center, CBCL Right.

|  | 3vs 8 | 8vs 3 | 1vs 7 | 9vs 4 | CBCL |
|---|---|---|---|---|---|
| **Spectral** | $0.8371 \pm 0.0056$ | $0.7830 \pm 0.0026$ | $0.9921 \pm 4.7283e - 04$ | $0.8651 \pm 0.0024$ | $0.8682 \pm 0.0023$ |
| **Parzen** | $0.7841 \pm 0.0069$ | $0.7656 \pm 0.0029$ | $0.9811 \pm 3.4158e - 04$ | $0.0.7244 \pm 0.0030$ | $0.8778 \pm 0.0023$ |
| **1CSVM** | $0.7896 \pm 0.0061$ | $0.7642 \pm 0.0032$ | $0.9889 \pm 1.8479e - 04$ | $0.7535 \pm 0.0041$ | $0.8824 \pm 0.0020$ |

Table 1: Average and standard deviation of the AUC for the different estimators on the considered tasks.

to derive finite sample bounds, to develop strategies to scale-up the algorithms to massive data-sets and to a more extensive experimental analysis.

## Footnotes

[1]Considering complex valued RKHS allows to use the theory of Fourier transform and for practical problems we can simply consider real valued kernels.

[2]Given an open subset $U$ and a compact subset $C \subset U$, a bump function is a continuous compactly supported function which is one on $C$ and its support is contained in $U$.

# References

[1] P. M. Anselone. *Collectively compact operator approximation theory and applications to integral equations*. Prentice-Hall Inc., Englewood Cliffs, N. J., 1971.

[2] N. Aronszajn. Theory of reproducing kernels. *Trans. Amer. Math. Soc.*, 68:337–404, 1950.

[3] M. Belkin, P. Niyogi, and V. Sindhwani. Manifold regularization: A geometric framework for learning from labeled and unlabeled examples. *J. Mach. Learn. Res.*, 7:2399–2434, 2006.

[4] C. Berg, J. Christensen, and P. Ressel. *Harmonic analysis on semigroups*, volume 100 of *Graduate Texts in Mathematics*. Springer-Verlag, New York, 1984.

[5] G. Biau, B. Cadre, D. Mason, and Bruno Pelletier. Asymptotic normality in density support estimation. *Electron. J. Probab.*, 14:no. 91, 2617–2635, 2009.

[6] S. Canu, Y. Grandvalet, V. Guigue, and A. Rakotomamonjy. Svm and kernel methods matlab toolbox. Perception Systmes et Information, INSA de Rouen, Rouen, France, 2005.

[7] C. Carmeli, E. De Vito, and A. Toigo. Vector valued reproducing kernel Hilbert spaces of integrable functions and Mercer theorem. *Anal. Appl. (Singap.)*, 4(4):377–408, 2006.

[8] V. Chandola, A. Banerjee, and V. Kumar. Anomaly detection: A survey. *ACM Comput. Surv.*, 41(3):1–58, 2009.

[9] H. W. Engl, M. Hanke, and A. Neubauer. *Regularization of inverse problems*, volume 375 of *Mathematics and its Applications*. Kluwer Academic Publishers Group, Dordrecht, 1996.

[10] M. Hein, O. Bousquet, and B. Schlkopf. Maximal margin classification for metric spaces. *Journal of Computer and System Sciences*, 71(3):333–359, 10 2005.

[11] H. Hoffmann. Kernel pca for novelty detection. *Pattern Recogn.*, 40(3):863–874, 2007.

[12] P Niyogi, S Smale, and S Weinberger. A topological view of unsupervised learning from noisy data. *preprint*, Jan 2008.

[13] R. Rifkin and R. Lippert. Notes on regularized least squares. Technical report, Massachusetts Institute of Technology, 2007.

[14] L. Rosasco, M. Belkin, and E. De Vito. On learning with integral operators. *J. Mach. Learn. Res.*, 11:905–934, 2010.

[15] S Roweis and L Saul. Nonlinear dimensionality reduction by locally linear embedding. *Science*, Jan 2000.

[16] B. Schölkopf, J. Giesen, and S. Spalinger. Kernel methods for implicit surface modeling. In *Advances in Neural Information Processing Systems 17*, pages 1193–1200, Cambridge, MA, 2005. MIT Press.

[17] B. Schölkopf, J. Platt, J. Shawe-Taylor, A. Smola, and R. Williamson. Estimating the support of a high-dimensional distribution. *Neural Comput.*, 13(7):1443–1471, 2001.

[18] S. Smale and D.X. Zhou. Geometry of probability spaces. *Constr. Approx.*, 30(3):311–323, 2009.

[19] I. Steinwart and A. Christmann. *Support vector machines*. Information Science and Statistics. Springer, New York, 2008.

[20] I. Steinwart, D. Hush, and C. Scovel. A classification framework for anomaly detection. *J. Mach. Learn. Res.*, 6:211–232 (electronic), 2005.

[21] J. Tenenbaum, V. Silva, and J. Langford. A global geometric framework for nonlinear dimensionality reduction. *Science*, Jan 2000.

[22] A. B. Tsybakov. On nonparametric estimation of density level sets. *Ann. Statist.*, 25(3):948–969, 1997.

[23] U. Von Luxburg. A tutorial on spectral clustering. *Statistics and Computing*, 17(4), 2007.

